# Bayesian Spike-Triggered Covariance Analysis

**Il Memming Park**
Center for Perceptual Systems
University of Texas at Austin
Austin, TX 78712, USA
memming@austin.utexas.edu

**Jonathan W. Pillow**
Center for Perceptual Systems
University of Texas at Austin
Austin, TX 78712, USA
pillow@mail.utexas.edu

## Abstract

Neurons typically respond to a restricted number of stimulus features within the high-dimensional space of natural stimuli. Here we describe an explicit model-based interpretation of traditional estimators for a neuron's multi-dimensional feature space, which allows for several important generalizations and extensions. First, we show that traditional estimators based on the spike-triggered average (STA) and spike-triggered covariance (STC) can be formalized in terms of the "expected log-likelihood" of a Linear-Nonlinear-Poisson (LNP) model with Gaussian stimuli. This model-based formulation allows us to define maximum-likelihood and Bayesian estimators that are statistically consistent and efficient in a wider variety of settings, such as with naturalistic (non-Gaussian) stimuli. It also allows us to employ Bayesian methods for regularization, smoothing, sparsification, and model comparison, and provides Bayesian confidence intervals on model parameters. We describe an empirical Bayes method for selecting the number of features, and extend the model to accommodate an arbitrary elliptical nonlinear response function, which results in a more powerful and more flexible model for feature space inference. We validate these methods using neural data recorded extracellularly from macaque primary visual cortex.

## 1 Introduction

A central problem in systems neuroscience is to understand the probabilistic relationship between sensory stimuli and neural responses. Most neurons in the early sensory pathway are only sensitive to a low-dimensional space of stimulus features, and ignore the other axes in the high-dimensional space of stimuli. Dimensionality reduction therefore plays an important role in neural characterization. The most popular dimensionality-reduction method for neural data uses the first two moments of the spike-triggered stimulus distribution: the spike-triggered average (STA) and the eigenvectors of the spike-triggered covariance (STC) [1–5]. These features are interpreted as filters or "receptive fields" that form the first stage in a linear-nonlinear-Poisson (LNP) cascade model [6, 7]. In this model, stimuli are projected onto a bank of linear filters, whose outputs are combined via a nonlinear function, which drives spiking as an inhomogeneous Poisson process (see Fig. 1).

Prior work has established the conditions for statistical consistency and efficiency of the STA and STC as feature space estimators [1, 2, 8, 9]. However, these moment-based estimators have not yet been interpreted in terms of an explicit probabilistic encoding model. We formalize that relationship here, building on a recent information-theoretic treatment of spike-triggered average and covariance analysis (iSTAC) [9]. Our general approach is inspired by probabilistic and Bayesian formulations of principal components analysis (PCA) and extreme components analysis (XCA), moment-based methods for linear dimensionality reduction that are closely related to STC analysis, but which were only more recently formulated in terms of an explicit probabilistic model [10–14].

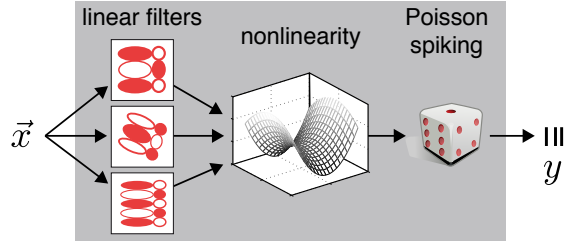

Figure 1: Schematic of linear-nonlinear-Poisson (LNP) neural encoding model [6].

Here we show, first of all, that STA and STC arise naturally from the *expected log-likelihood* of an LNP model with an "exponentiated-quadratic" nonlinearity, where expectation is taken with respect to a Gaussian stimulus distribution. This insight allows us to formulate exact maximum-likelihood estimators that apply to arbitrary stimulus distributions. We then introduce Bayesian methods for regularizing and smoothing receptive field estimates, and an approximate empirical Bayes method for selecting the feature space dimensionality, which obviates nested hypothesis tests, bootstrapping, or cross-validation based methods [5]. Finally, we generalize these estimators to accommodate LNP models with arbitrary elliptically symmetric nonlinearities. The resulting model class provides a richer and more flexible model of neural responses but can still recover a high-dimensional feature space (unlike more general information-theoretic estimators [8, 15], which do not scale easily to more than 2 filters). We apply these methods to a variety of simulated datasets and to responses from neurons in macaque primary visual cortex stimulated with binary white noise stimuli [16].

## 2 Model-based STA and STC

In a typical neural characterization experiment, the experimenter presents a train of rapidly varying sensory stimuli and records a spike train response. Let $\mathbf{x}$ denote a $D$-dimensional vector containing the spatio-temporal stimulus affecting a neuron's scalar spike response $y$ in a single time bin. A principal goal of neural characterization is to identify $\beta$, a low-dimensional projection matrix such that $\beta^\top \mathbf{x}$ captures the neuron's dependence on the stimulus $\mathbf{x}$. The columns of $\beta$ can be regarded as linear receptive fields that provide a basis for the neural feature space.

The methods we consider here all assume that neural responses can be described by an LNP cascade model (Fig. 1). Under this model, the conditional probability of a response $y|\mathbf{x}$ is Poisson with rate $f(\beta^\top \mathbf{x})$, where $f$ is a vector function mapping feature space to instantaneous spike rate.[1]

### 2.1 STA and STC analysis

The STA and the STC matrix are the (empirical) first and second moments, respectively, of the spike-triggered stimulus ensemble $\{\mathbf{x}_i | y_i\}_{i=1}^N$. They are defined as:

$$\text{STA: } \mu = \frac{1}{n_{sp}} \sum_{i=1}^N y_i \mathbf{x}_i, \quad \text{and} \quad \text{STC: } \Lambda = \frac{1}{n_{sp}} \sum_{i=1}^N y_i (\mathbf{x}_i - \mu)(\mathbf{x}_i - \mu)^\top, \tag{1}$$

where $n_{sp} = \sum y_i$ is the number of spikes and $N$ is the total number of time bins. Traditional STA/STC analysis provides an estimate for the feature space basis $\beta$ consisting of: **(1)** $\mu$, if it is significantly different from zero; and **(2)** the eigenvectors of $\Lambda$ whose eigenvalues are significantly smaller or larger from those of the prior stimulus covariance $\Phi = \mathbb{E}[\mathbf{x}\mathbf{x}^T]$. This estimate is provably consistent only in the case of stimuli drawn from a spherically symmetric (for STA) or independent Gaussian distribution (for STC) [17].[2]

## 2.2 Equivalent model-based formulation

Motivated by [9], we consider an LNP model where the spike rate is defined by an exponentiated general quadratic function:

$$f(\mathbf{x}) = \exp\left(\tfrac{1}{2}\mathbf{x}^\top C \mathbf{x} + b^\top \mathbf{x} + a\right), \tag{2}$$

where $C$ is a symmetric matrix, $b$ is a vector, and $a$ is a scalar. Then the log-likelihood per spike, the conditional log-probability of the data divided by the number of spikes, is

$$\mathcal{L} = \tfrac{1}{n_{sp}} \sum_i \log P(y_i | C, b, a, \mathbf{x}_i) = \tfrac{1}{n_{sp}} \sum_i \left(y_i \log f(\mathbf{x}_i) - f(\mathbf{x}_i)\right) \tag{3}$$

$$= \tfrac{1}{2}\operatorname{Tr}[C\Lambda] + \tfrac{1}{2}\mu^\top C\mu + b^\top \mu + a - \tfrac{N}{n_{sp}} e^a \left[\tfrac{1}{N}\sum_i \exp\left(\tfrac{1}{2}\mathbf{x}_i{}^\top C\mathbf{x}_i + b^\top \mathbf{x}_i\right)\right]. \tag{4}$$

If the stimuli are drawn from $\mathbf{x} \sim \mathcal{N}(0, \Phi)$, a zero-mean Gaussian with covariance $\Phi$, then the expression in square brackets (eq. 4) will converge to its expectation, given by:

$$\mathbb{E}\left[e^{\tfrac{1}{2}\mathbf{x}^\top C\mathbf{x} + b^\top \mathbf{x}}\right] = |I - \Phi C|^{-\tfrac{1}{2}} \exp\left(\tfrac{1}{2}b^\top (\Phi^{-1} - C)^{-1} b\right), \tag{5}$$

so long as $(\Phi^{-1} - C)$ is invertible and positive definite.[3] Substituting this expectation (eq. 5) into the log-likelihood (eq. 4) yields a quantity we call the *expected log-likelihood* $\tilde{\mathcal{L}}$, which can be expressed in terms of the STA, STC, $\Phi$, and the model parameters:

$$\tilde{\mathcal{L}} = \tfrac{1}{2}\operatorname{Tr}[C\Lambda] + \tfrac{1}{2}\mu^\top C\mu + b^\top \mu + a - \tfrac{N}{n_{sp}} |I - \Phi C|^{-\tfrac{1}{2}} \exp\left(\tfrac{1}{2}b^\top (\Phi^{-1} - C)^{-1} b + a\right). \tag{6}$$

Maximizing this expression yields expected-ML estimates (see online supplement for derivation):

$$\tilde{C}_{\mathrm{ml}} = \Phi^{-1} - \Lambda^{-1}, \qquad \tilde{b}_{\mathrm{ml}} = \Lambda^{-1}\mu,$$

$$\tilde{a}_{\mathrm{ml}} = \log\left(\tfrac{n_{sp}}{N}\left|\Phi\Lambda^{-1}\right|^{\tfrac{1}{2}}\right) - \tfrac{1}{2}\mu^\top \Phi^{-1}\Lambda^{-1}\mu. \tag{7}$$

Thus, for an LNP model with exponentiated-quadratic nonlinearity stimulated with Gaussian noise, the (expected) maximum likelihood estimates can be obtained in closed form from the STA, STC, stimulus covariance, and mean spike rate $n_{sp}/N$.

Several features of this solution are worth remarking. First, if the quadratic component $C = 0$, then $\tilde{b}_{\mathrm{ml}} = \Phi^{-1}\mu$, the whitened STA (as in [17]). Second, if the stimuli are white, meaning $\Phi = I$, then $\tilde{C}_{\mathrm{ml}} = I - \Lambda^{-1}$, which has the same eigenvectors as the STC matrix. Third, if we plug the expected-ML estimates back into the log-likelihood, we get

$$\tilde{\mathcal{L}} = \tfrac{1}{2}\left(\operatorname{Tr}\left[\Lambda\Phi^{-1}\right] + \mu^\top \Phi^{-1}\mu - \log\left|\Lambda\Phi^{-1}\right|\right) + const \tag{8}$$

which (for $\Phi = I$) is the information-theoretic spike-triggered average and covariance (iSTAC) cost function [9]. The iSTAC estimator finds the subspace that maximizes the "single-spike information" [18] under a Gaussian model of the raw and spike-triggered stimulus distributions (that coincides with (eq. 8)), but its precise relationship to maximum likelihood has not been shown previously.

## 2.3 Generalizing to non-Gaussian stimuli

The conditions for which the STA and STC provide asymptotically efficient estimators for a neural feature space are clear from the derivations above: if the stimuli are Gaussian (a condition which is rarely if ever met in practice), the STA is optimal when the nonlinearity is $f(\mathbf{x}) = \exp(b^\top \mathbf{x} + a)$ (as shown in [8]); the STC is optimal when $f(\mathbf{x}) = \exp(\mathbf{x}^\top C\mathbf{x} + a)$ (as shown in [9]).

However, the maximum of the *exact* model log-likelihood (eq. 4) yields a consistent and asymptotically efficient estimator even when stimuli are *not* Gaussian. Numerically optimizing this loss

function is computationally more expensive than computing the STA and STC eigendecomposition, but the log-likelihood is jointly concave in the model parameters $(C, b, a)$, meaning ML estimates can be obtained rapidly by convex optimization [19].

For cases where $\mathbf{x}$ is high-dimensional, it is easier to directly estimate a low-rank representation of $C$, rather than optimize the entire $D \times D$ matrix. We therefore define a rank-$d$ representation for $C$:

$$C = \sum_{i=1}^{d} \mathbf{w}_i s_i \mathbf{w}_i^\top = WSW^\top, \tag{9}$$

where $W$ is a matrix whose columns $\mathbf{w}_i$ are features, $s_i \in \{-1, 1\}$ are constants that control the shape of the nonlinearity along each axis in feature space (-1 for suppressive, +1 for excitatory), and $S$ is a diagonal matrix containing $s_i$ along the diagonal. (We will assume the $s_i$ are fixed using the sign of the eigenvalues of the expected-ML estimate $\tilde{C}_{\mathrm{ml}}$, and not varied thereafter).

The feature space of the resulting model is spanned by $b$ and the columns of $W$. We refer to ML estimators for $(b, W)$ as maximum-likelihood STA and STC (or *exact ML*, as opposed to *expected-ML* estimates from moment-based formulas (eq. 7); see Figs. 2-3 for comparisons). These estimates will closely match the standard STA and STC-based feature space when stimuli are Gaussian, but (as maximum-likelihood estimates) are also consistent and asymptotically efficient for arbitrary stimuli.

An additional difference between maximum-likelihood and standard STA/STC analysis is that the parameters $(b, W)$ have meaningful units of length: the vector norm of $b$ determines the amplitude of the "linear" contribution to the neural response (via $b^\top \mathbf{x}$), while the norm of columns in $W$ determines the amplitude of "symmetric" excitatory or suppressive contributions to the response (via $\mathbf{x}^\top WSW^\top \mathbf{x}$). Shrinking these vectors (e.g., with a prior) has the effect of reducing their influence in the model, and they drop out of the model entirely if we shrink them to zero (a fact that we will exploit in the next section). By contrast, the standard STA and STC eigenvectors are usually taken as unit vectors, providing a basis for the neural feature space in which the nonlinearity ("N" stage) must still be estimated. We are free to normalize the ML estimates $(\hat{b}, \hat{W})$ and estimate an arbitrary nonlinearity in a similar manner, but it is noteworthy that the parameters $(a, b, W)$ specify a complete encoding model in and of themselves.

# 3   Bayesian STC

Now that we have defined an explicit model and likelihood function underlying STA and STC analysis, we can straightforwardly apply Bayesian methods for estimation, prediction, error bars, model comparison, etc., by introducing a prior over the model parameters. Bayesian methods can be especially useful in cases where we have prior information (e.g., about smoothness or sparseness of neural features, [20–25]), and in general have attractive theoretical properties for high-dimensional inference problems [26–28].

Here we consider two types of priors: (1) a smoothing prior, which holds the filters to be smooth in space/time; and (2) a sparsifying prior, which we employ to directly estimate the feature space dimensionality (i.e., the number of significant filters). We apply these priors to $b$ and the columns of $W$, in conjunction with either exact (for accuracy) or expected (for speed) log-likelihood functions defined above. We refer to the resulting estimators as *Bayesian STC* (or "BSTC").

We perform BSTC estimation by maximizing the sum of log-likelihood and log-prior to obtain *maximum a posteriori* (MAP) estimates of the filters and constant $a$. It is worth noting that since the derivatives of the expected likelihood (eq. 6) are also written in terms of STA/STC, optimization using the expected log-likelihood can be carried out more efficiently—it reduces the cost of each iteration by a factor of $N$ compared to optimizing the exact likelihood (eq. 3).

## 3.1   Smoothing prior

Neural receptive fields are generally smooth, so a prior that encourages this tendency will tend to improve performance. Receptive field estimates under such a prior will be smooth unless the likelihood provides sufficient evidence for jaggedness. To encourage smoothness, we placed a zero-mean Gaussian prior on the second-order differences of each filter [29]:

$$L\mathbf{w} \sim \mathcal{N}(0, \phi^{-1}I), \tag{10}$$

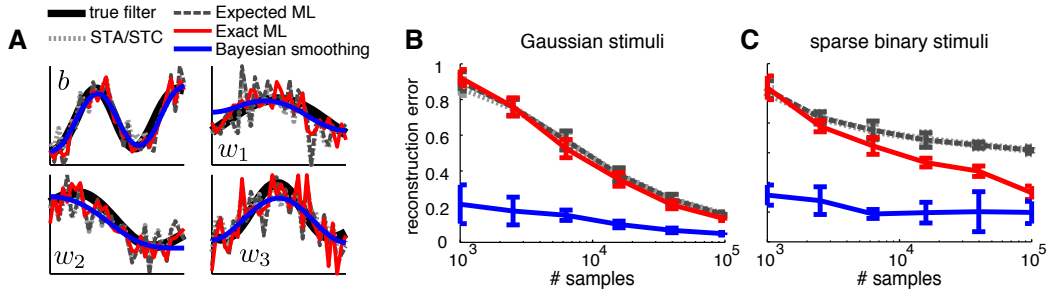

Figure 2: Estimated filters and error rates for various estimators. An LNP model with 4 orthogonal 32-elements filters (see text) was simulated with two types of stimuli (A-B: white Gaussian: C: sparse binary). Mean firing rate 0.16 spk/s. **(A)** Filters estimated from 10,000 samples. STA/STC filters are normalized to match the norm of true filters. **(B)** Convergence to the true filter under each method, Gaussian stimuli. **(C)** Convergence for sparse binary stimuli.

where $L$ is the discrete Laplacian operator and $\phi$ is a hyperparameter controlling the smoothness of feature vectors. This is equivalent to imposing a penalty (given by $\frac{1}{2}\phi\mathbf{w}_i^\top LL^\top\mathbf{w}_i$) on the squared second derivatives $b$ and $W$ in the optimization function. Larger $\phi$ implies a narrower Gaussian prior on these differences, hence a stronger preference for smooth filters. For simplicity, we assumed all filters came from the same prior, resulting in a single hyperparameter $\phi$ for all filters, and used cross-validation to choose an appropriate $\phi$ for each dataset.

To illustrate the effects of this prior, we simulated an example dataset from an LNP neuron with exponentiated-quadratic nonlinearity and four 32-element, 1-dimensional (temporal) filters. The filter shapes were given by orthogonalized randomly-placed Gaussians (Fig. 2). We fixed the dimensionality of our feature space estimates to be the same as the true model, since our focus was the quality of each corresponding filter estimate.

For Gaussian stimuli, we found that classical STA/STC, expected-ML, and exact-ML estimates were indistinguishable (Fig. 2). However, for "sparse" binary stimuli (3 of the 32 pixels set randomly to $\pm 1$), for which STA/STC and expected-ML estimates are no longer consistent, we found significantly better performance from the exact-ML estimates (Fig. 2C). Most importantly, for both Gaussian and sparse stimuli alike, the smoothing prior provided a large improvement in the quality of feature space estimates, achieving similar error with 2 orders of magnitude fewer stimuli.

### 3.2 Automatic selection of feature space dimensionality

While smoothing regularizes receptive field estimates by penalizing filter roughness, a perhaps more critical aspect of the STA/STC model is its vast number of possible parameters due to uncertainty in the number of filters. Our approach to this problem was inspired by Bayesian PCA [10], a method for automatically choosing the number of meaningful principle components using a "feature-selection prior" designed to encourage sparsity. The basic idea behind this approach is that a zero-mean Gaussian prior on each filter $\mathbf{w}_i$ (separately controlled by a hyperparameter $\alpha_i$) can be used to "shrink to zero" any components that do not contribute meaningfully to the evidence, just as in *automatic relevance determination* (ARD), also known as *sparse Bayesian learning* [27,30]. Unlike PCA, we seek to preserve components of the STC matrix with both large and small eigenvalues, which correspond to excitatory and suppressive filters, respectively. One solution to this problem, Bayesian Extreme Components Analysis [14], preserves large and small eigenvalues of the covariance matrix, but does not incorporate additional priors on filter shape, and has not yet been formulated for our (Poisson) likelihood function. Instead, we address the problem by using the sign of the diagonal elements in $S$ to determine whether a feature $\mathbf{w}$ produces a positive or negative eigenvalue in $C$ (eq. 9). (Recall that the eigenvalues of $C = \Phi^{-1} - \Lambda^{-1}$ are positive and negative, while those of the STC matrix $\Lambda$ are strictly positive). Reparametrizing the STC in terms of $C$ therefore allows us to apply a variant of the Bayesian PCA algorithm directly to $b$ and the columns of $W$.

The details of our approach are as follows. We put the ARD prior on each column of $W$:

$$\mathbf{w}_i \sim \mathcal{N}\left(0, \alpha_i^{-1}I\right), \tag{11}$$

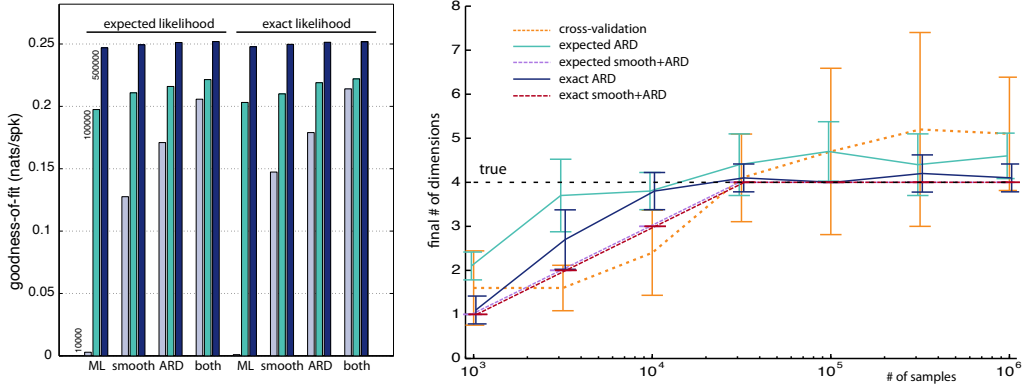

Figure 3: Goodness-of-fit of estimated models and the estimated dimension as a function of number of samples. The same simulation parameters as Fig. 2 were used. **Left:** Information per spike (normalized difference in log-likelihoods) captured by different estimates. Models were estimated from $10^3$, $10^4$, and $5 \times 10^4$ stimuli respectively. **Right:** Estimated number of dimensions as a function of the number of training samples. When both smoothing and ARD priors are used, the variability rapidly diminishes to near zero.

where $\alpha_i$ is a hyperparameter controlling the prior variance of $\mathbf{w}_i$. We impose the same prior on $b$, with an additional hyperparamter $\alpha_0$, resulting in $(D+1)$ hyperparameters for the complete model. We initialize $b$ to its ML estimate and the $\mathbf{w}_i$ to the eigenvectors of $\tilde{C}_{\mathrm{ml}}$, scaled by the square root of their eigenvalues. Then, we optimize the parameters and hyperparameters in a similar fashion to the Bayesian PCA algorithm [10]: we alternate between maximizing the posterior for the parameters $(a, b, W)$ given hyperparameters $\alpha$, and evidence optimization ($\arg\max_\alpha \Pr[(\mathbf{x}, y)|\alpha]$) to update $\alpha$. Since a closed form for the evidence is not known, we use the approximate fixed point update rule developed in [10]: $\alpha_i^{new} \leftarrow \frac{D}{|||w_i||^2}$. This update is valid when each element of the receptive field $\mathbf{w}_i$ is well defined (non-zero), otherwise it overestimates the corresponding $\alpha_i$. The algorithm begins with all $\alpha_i$ set to zero (infinite prior variance), giving ML estimates for the parameters. Subsequent updates will cause some $\alpha_i$ to grow without bound, shrinking the prior variance of the corresponding feature vector $\mathbf{w}_i$ until it drops out of the model entirely as $\alpha_i \to \infty$. The remaining $\mathbf{w}_j$, for which $\alpha_j$ remain finite, define the feature space estimate. Note that these updates are fast (especially with expected log-likelihood), providing a much less computationally intensive estimate of feature space dimensionality than bootstrap-based methods [5].

Figure 3 (left) shows that ARD prior greatly increases the model goodness-of-fit (likelihood on test data), and is synergistic with the smoothing prior defined above. The improvement (relative to ML estimates) is greatest when the number of samples is small, and it enhances both expected and exact likelihood estimates. We compared this method for estimating feature space dimensionality with a more classical (non-Bayesian) approach based on cross-validation. We first fit a full-rank model with exact likelihood, and built a sparse model by adding filters from this set greedily until the likelihood of test data began to decrease. The resulting estimate of dimension is underestimated when there is not enough data, and even with large amount of data, it has high variance (Fig. 3, right). In comparison, our ARD-based estimate converged quickly to the correct dimension and exhibited smaller variability. When both smoothing and ARD priors were used, the variability decreased markedly and always achieved the correct dimension even for moderate amounts of data. One additional advantage of Bayesian approach is that it can use all the available data; under cross-validation, some proportion of data is needed to form the test set (in this example we provided extra data for this method only).

## 4   Extension: the elliptical-LNP model

Finally, the model and inference procedures we have described above can be extended to a much more general class of response functions with zero additional computational cost. We can replace the exponential function which operates on the quadratic form in the model nonlinearity (eq. 2)

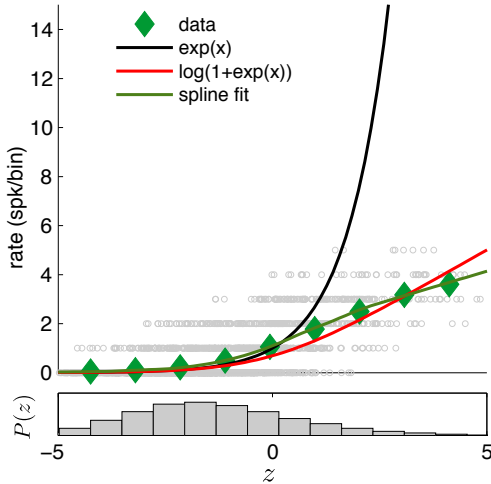

Figure 4: 1-D nonlinear functions $g$ mapping $z$, the output of the quadratic stage, to spike rate for a V1 complex cell [16]. The exact-ML filter estimate for $W$ and $b$ were obtained using the smoothing BSTC with an exponential nonlinearity. (Final filter estimates for this cell shown in Fig. 5). The quadratic projection ($z$) was computed using the filter estimates, and is plotted against the observed spike counts (gray circles), histogram-based estimate of the nonlinearity (green diamonds), exponential nonlinearity (black trace), a well-known alternative nonlinearity $\log(1 + e^z)$ (red), and a cubic spline estimated using 7 knots (green trace). We fixed the fitted cubic spline nonlinearity and then refit the filters, resulting in an estimate of the elliptical-LNP model.

with an arbitrary function $g(\cdot)$, resulting in a model class that includes any elliptically symmetric mapping of the stimulus to spike rate. We call this the *elliptical-LNP model*.

The elliptical-LNP model can be formalized by writing the nonlinearity $f(\mathbf{x})$ (depicted in Fig. 1) as the composition of two nonlinear functions: a quadratic function that maps high dimensional stimulus to real line $z(\mathbf{x}) = \frac{1}{2}\mathbf{x}^\top C\mathbf{x} + b^\top \mathbf{x} + a$, and a 1-D nonlinearity $g(z)$. The full nonlinearity is thus $f(\mathbf{x}) = g(z(\mathbf{x}))$.

Although LNP with exponential nonlinearity has been widely adapted in neuroscience for its simplicity, the actual nonlinearity of neural systems is often sub-exponential. Moreover, the effect of nonlinearity is even more pronounced in the exponentiated-quadratic function, and hence it may be helpful to use a sub-exponential function $g$. Figure 4 shows the nonlinearity of an example neuron from V1 (see next section) compared to $g(z) = e^z$ (the assumption implicit in STA/STC), a more linear function $g(z) = \log(1 + e^z)$, and a cubic spline fit by maximum likelihood.

The likelihood given by eq. 3 can be optimized efficiently as long as $g$ and $g'$ can be computed efficiently. The log-likelihood is concave in $(a, b, C)$ so long as $g$ obeys the standard regularity conditions (convex and log-concave), but we did not impose those conditions here. For fast optimization, we first used the exponentiated-quadratic nonlinearity as an initialization (expected then exact-ML), then we refined the model with a spline nonlinearity.

## 5 Application to neural data

We applied BSTC to data from a V1 complex cell (data published in [16]). The stimulus consisted of oriented binary white noise ("flickering bars") aligned with the cell's preferred orientation. We selected a cell (544l029.p21) that was reported to have large set of filters, to illustrate the power of our technique. The size of receptive field was chosen to be 16 bars $\times$ 10 time bins, yielding a 160-dimensional stimulus space. Three features of this data that make BSTC appropriate: (1) the stimulus is non-Gaussian; (2) the nonlinearity is not exponential (Fig. 4); (3) the filters are smooth in space and time (Fig. 5).

We estimated the nonlinearity using a cubic spline, and applied a smoothing BSTC to $10^4$ samples presented at 100 Hz (Fig. 5, top). The ARD-prior BSTC estimate trained on $2\times10^5$ stimuli preserved 14 filters (Fig. 5, bottom). The quality of the filters are qualitatively close to that obtained by STA/STC. However, the resulting model has better overall goodness-of-fit, as well as significant improvement over the exact ML model for each reduced dimension model (Fig. 6). To achieve the same level of fit as using 2 filters for BSTC, the exact ML based sparse model required 6 additional filters (dotted line).

We also compared BSTC to a generalized linear model (GLM) with same number of linear and quadratic filters fit by STA/STC (a method described previously by [7]). This approach places a prior over the weights on squared filter outputs, but not on the filters themselves. On a test set,

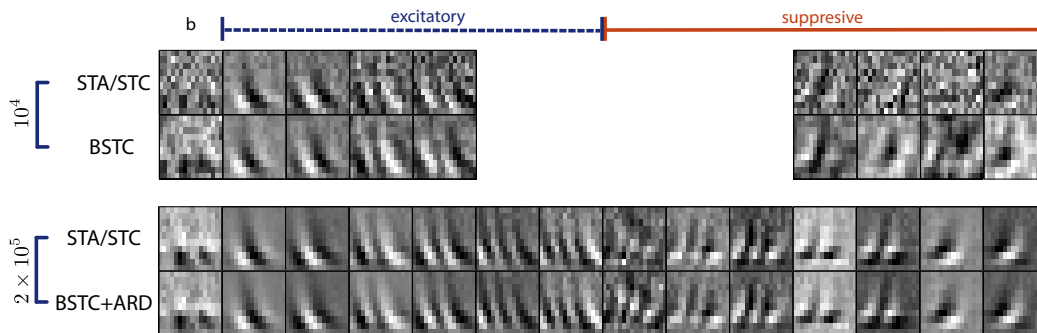

Figure 5: Estimating visual receptive fields from a complex cell. Each image corresponds to a normalized 16 dimensions spatial pixels (horizontal) by 10 time bins (vertical) filter. (top) Smoothing prior recovers better filters. Bayesian STC (BSTC) with smoothing prior and fixed spline nonlinearity applied to a fixed number of filters. (bottom) Sparsification determines the number of filters. BSTC with ARD, smoothing, and spline nonlinearity recovers 14 receptive fields out of 160.

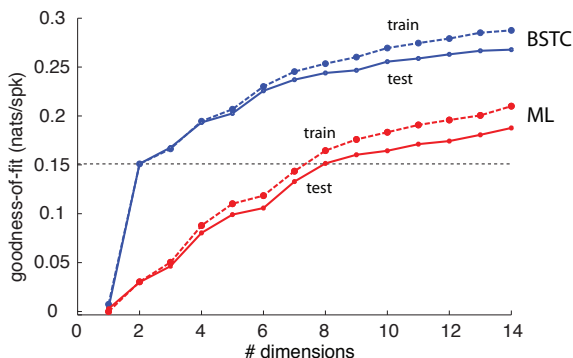

Figure 6: Goodness-of-model fits from exact ML solution with exponential nonlinearity compared to BSTC with a fixed spline nonlinearity and smoothing prior ($2 \times 10^5$ samples). Filters are added in the order that increases the likelihood on the training set the most. The corresponding filters are visualized in fig. 5.

BSTC outperformed the GLM on all cells in the dataset, achieving 34% more bits/spike (normalized log-likelihood) over a population of 50 cells.

## 6 Conclusion

We have provided an explicit, probabilistic, model-based framework that formalizes the classical moment-based estimators (STA, STC) and a more recent information-theoretic estimator (iSTAC) for neural feature spaces. The maximum of the "expected log-likelihood" under this model, where expectation is taken with respect to Gaussian stimulus distribution, corresponds precisely to the moment-based estimators for uncorrelated stimuli. A model-based formulation allows us to compute exact maximum-likelihood estimates when stimuli are non-Gaussian, and we have incorporated priors in conjunction with both expected and exact likelihoods to achieve Bayesian methods for smoothing and feature selection (estimation of the number of filters).

The elliptical-LNP model extends BSTC analysis to a richer class of nonlinear response models. Although the assumption of elliptical symmetry makes it less general than information-theoretic estimators such as *maximally informative dimensions* (MID) [8, 15], it has significant advantages in computational efficiency, number of local optima, and suitability for high-dimensional feature spaces. The elliptical-LNP model may also be easily extended to incorporate spike-history effects by adding linear projections of the neuron's spike history as inputs, as in the generalized linear model (GLM) [9, 17, 25, 31]. We feel the synthesis of multi-dimensional nonlinear stimulus sensitivity (as described here) and non-Poisson, history-dependent spiking presents a promising tool for unlocking the statistical structure of the neural code.

## Footnotes

[1] Here $f$ has units of spikes/bin, for some fixed bin size $\triangle$. In the limit $\triangle \to 0$, the model output is an inhomogeneous Poisson process, but we use discrete time bins here for concreteness.

[2] For elliptically symmetric or colored Gaussian stimuli, a consistent estimate requires whitening the stimuli by $\Phi^{-\frac{1}{2}}$ and then multiplying the estimated features (STA and STC eigenvectors) again by $\Phi^{-\frac{1}{2}}$ (see [5]).

[3]If it is not, then this expectation does not exist, and simulations of the corresponding model will produce impossibly high spike counts, with STA and STC dominated by the response to a single stimulus.

# References

[1] J. Bussgang. Crosscorrelation functions of amplitude-distorted gaussian signals. *RLE Technical Reports*, 216, 1952.

[2] E. J. Chichilnisky. A simple white noise analysis of neuronal light responses. *Network: Comput. Neural Syst.*, 12:199–213, 2001.

[3] R. de Ruyter and W. Bialek. Real-time performance of a movement-senstive neuron in the blowfly visual system. *Proc. R. Soc. Lond. B*, 234:379–414, 1988.

[4] O. Schwartz, E. J. Chichilnisky, and E. P. Simoncelli. Characterizing neural gain control using spike-triggered covariance. *Adv. Neural Information Processing Systems*, pages 269–276, 2002.

[5] O. Schwartz, J. W. Pillow, N. C. Rust, and E. P. Simoncelli. Spike-triggered neural characterization. *J. Vision*, 6(4):484–507, 7 2006.

[6] E. P. Simoncelli, J. Pillow, L. Paninski, and O. Schwartz. Characterization of neural responses with stochastic stimuli. *The Cognitive Neurosciences, III*, chapter 23, pages 327–338. MIT Press, 2004.

[7] S. Gerwinn, J. Macke, M. Seeger, and M. Bethge. Bayesian inference for spiking neuron models with a sparsity prior. *Adv. in Neural Information Processing Systems 20*, pages 529–536. MIT Press, 2008.

[8] L. Paninski. Convergence properties of some spike-triggered analysis techniques. *Network: Comput. Neural Syst.*, 14:437–464, 2003.

[9] J. W. Pillow and E. P. Simoncelli. Dimensionality reduction in neural models: An information-theoretic generalization of spike-triggered average and covariance analysis. *J. Vision*, 6(4):414–428, 4 2006.

[10] C. M. Bishop. Bayesian PCA. *Adv. in Neural Information Processing Systems*, pages 382–388, 1999.

[11] M. E. Tipping and C. M. Bishop. Probabilistic principal component analysis. *J. the Royal Statistical Society. Series B, Statistical Methodology*, pages 611–622, 1999.

[12] T. P. Minka. Automatic choice of dimensionality for PCA. *NIPS*, pages 598–604, 2001.

[13] M. Welling, F. Agakov, and C. K. I. Williams. Extreme components analysis. *Adv. in Neural Information Processing Systems 16*. MIT Press, 2004.

[14] Y. Chen and M. Welling. Bayesian extreme components analysis. *IJCAI*, 2009.

[15] T. Sharpee, N. C. Rust, and W. Bialek. Analyzing neural responses to natural signals: maximally informative dimensions. *Neural Comput*, 16(2):223–250, Feb 2004.

[16] N. C. Rust, O. Schwartz, J. A. Movshon, and E. P. Simoncelli. Spatiotemporal elements of macaque V1 receptive fields. *Neuron*, 46(6):945–956, Jun 2005.

[17] L. Paninski. Maximum likelihood estimation of cascade point-process neural encoding models. *Network: Comput. Neural Syst.*, 15(04):243–262, November 2004.

[18] N. Brenner, S. P. Strong, R. Koberle, W. Bialek, and R. R. de Ruyter van Steveninck. Synergy in a neural code. *Neural Comput*, 12(7):1531–1552, Jul 2000.

[19] L. Paninski. Maximum likelihood estimation of cascade point-process neural encoding models. *Network: Computation in Neural Systems*, 15:243–262, 2004.

[20] F. Theunissen, S. David, N. Singh, A. Hsu, W. Vinje, and J. Gallant. Estimating spatio-temporal receptive fields of auditory and visual neurons from their responses to natural stimuli. *Network: Comput. Neural Syst.*, 12:289–316, 2001.

[21] M. Sahani and J. Linden. Evidence optimization techniques for estimating stimulus-response functions. *NIPS*, 15, 2003.

[22] S. V. David, N. Mesgarani, and S. A. Shamma. Estimating sparse spectro-temporal receptive fields with natural stimuli. *Network: Comput. Neural Syst.*, 18(3):191–212, 2007.

[23] I. H. Stevenson, J. M. Rebesco, N. G. Hatsopoulos, Z. Haga, L. E. Miller, and K. P. Körding. Bayesian inference of functional connectivity and network structure from spikes. *IEEE Transactions on Neural Systems and Rehabilitation Engineering*, 17(3):203–213, 2009.

[24] S. Gerwinn, J. H Macke, and M. Bethge. Bayesian inference for generalized linear models for spiking neurons. *Frontiers in Computational Neuroscience*, 2010.

[25] A. Calabrese, J. W. Schumacher, D. M. Schneider, L. Paninski, and S. M. N. Woolley. A generalized linear model for estimating spectrotemporal receptive fields from responses to natural sounds. *PLoS One*, 6(1):e16104, 2011.

[26] W. James and C. Stein. Estimation with quadratic loss. *4th Berkeley Symposium on Mathematical Statistics and Probability*, 1:361–379, 1960.

[27] M. Tipping. Sparse Bayesian learning and the relevance vector machine. *JMLR*, 1:211–244, 2001.

[28] D. Donoho and M. Elad. Optimally sparse representation in general (nonorthogonal) dictionaries via $l^1$ minimization. *PNAS*, 100:2197–2202, 2003.

[29] K. R. Rad and L. Paninski. Efficient, adaptive estimation of two-dimensional firing rate surfaces via gaussian process methods. *Network: Comput. Neural Syst.*, 21(3-4):142–168, 2010.

[30] D. Wipf and S. Nagarajan. A new view of automatic relevance determination. *Adv. in Neural Information Processing Systems 20*, pages 1625–1632. MIT Press, 2008.

[31] W. Truccolo, U. T. Eden, M. R. Fellows, J. P. Donoghue, and E. N. Brown. A point process framework for relating neural spiking activity to spiking history, neural ensemble and extrinsic covariate effects. *J. Neurophysiol*, 93(2):1074–1089, 2005.

